# VIBES: A Variational Inference Engine for Bayesian Networks

**Christopher M. Bishop**
Microsoft Research
Cambridge, CB3 0FB, U.K.
research.microsoft.com/~cmbishop

**David Spiegelhalter**
MRC Biostatistics Unit
Cambridge, U.K.
*david.spiegelhalter@mrc-bsu.cam.ac.uk*

**John Winn**
Department of Physics
University of Cambridge, U.K.
www.inference.phy.cam.ac.uk/jmw39

## Abstract

In recent years variational methods have become a popular tool for approximate inference and learning in a wide variety of probabilistic models. For each new application, however, it is currently necessary first to derive the variational update equations, and then to implement them in application-specific code. Each of these steps is both time consuming and error prone. In this paper we describe a general purpose inference engine called VIBES ('Variational Inference for Bayesian Networks') which allows a wide variety of probabilistic models to be implemented and solved variationally without recourse to coding. New models are specified either through a simple script or via a graphical interface analogous to a drawing package. VIBES then automatically generates and solves the variational equations. We illustrate the power and flexibility of VIBES using examples from Bayesian mixture modelling.

## 1 Introduction

Variational methods [1, 2] have been used successfully for a wide range of models, and new applications are constantly being explored. In many ways the variational framework can be seen as a complementary approach to that of Markov chain Monte Carlo (MCMC), with different strengths and weaknesses.

For many years there has existed a powerful tool for tackling new problems using MCMC, called BUGS ('Bayesian inference Using Gibbs Sampling') [3]. In BUGS a new probabilistic model, expressed as a directed acyclic graph, can be encoded using a simple scripting notation, and then samples can be drawn from the posterior distribution (given some data set of observed values) using Gibbs sampling in a way that is largely automatic. Furthermore, an extension called WinBUGS provides a graphical front end to BUGS in which the user draws a pictorial representation of

the directed graph, and this automatically generates the required script.

We have been inspired by the success of BUGS to produce an analogous tool for the solution of problems using variational methods. The challenge is to build a system that can handle a wide range of graph structures, a broad variety of common conditional probability distributions at the nodes, and a range of variational approximating distributions. All of this must be achieved whilst also remaining computationally efficient.

## 2 A General Framework for Variational Inference

In this section we briefly review the variational framework, and then we characterise a large class of models for which the variational method can be implemented automatically. We denote the set of all variables in the model by $W = (V, X)$ where $V$ are the visible (observed) variables and $X$ are the hidden (latent) variables. As with BUGS, we focus on models that are specified in terms of an acyclic directed graph (treatment of undirected graphical models is equally possible and is somewhat more straightforward). The joint distribution $P(V, X)$ is then expressed in terms of conditional distributions $P(W_i|\text{pa}_i)$ at each node $i$, where $\text{pa}_i$ denotes the set of variables corresponding to the parents of node $i$, and $W_i$ denotes the variable, or group of variables, associated with node $i$. The joint distribution of all variables is then given by the product of the conditionals $P(V, X) = \prod_i P(W_i|\text{pa}_i)$.

Our goal is to find a variational distribution $Q(X|V)$ that approximates the true posterior distribution $P(X|V)$. To do this we note the following decomposition of the log marginal probability of the observed data, which holds for any choice of distribution $Q(X|V)$

$$\ln P(V) = \mathcal{L}(Q) + \text{KL}(Q\|P) \tag{1}$$

where

$$\mathcal{L}(Q) = \sum_X Q(X|V) \ln \frac{P(V, X)}{Q(X|V)} \tag{2}$$

$$\text{KL}(Q\|P) = -\sum_X Q(X|V) \ln \frac{P(X|V)}{Q(X|V)} \tag{3}$$

and the sums are replaced by integrals in the case of continuous variables. Here $\text{KL}(Q\|P)$ is the Kullback-Leibler divergence between the variational approximation $Q(X|V)$ and the true posterior $P(X|V)$. Since this satisfies $\text{KL}(Q\|P) \geq 0$ it follows from (1) that the quantity $\mathcal{L}(Q)$ forms a lower bound on $\ln P(V)$.

We now choose some family of distributions to represent $Q(X|V)$ and then seek a member of that family that maximizes the lower bound $\mathcal{L}(Q)$. If we allow $Q(X|V)$ to have complete flexibility then we see that the maximum of the lower bound occurs for $Q(X|V) = P(X|V)$ so that the variational posterior distribution equals the true posterior. In this case the Kullback-Leibler divergence vanishes and $\mathcal{L}(Q) = \ln P(V)$. However, working with the true posterior distribution is computationally intractable (otherwise we wouldn't be resorting to variational methods). We must therefore consider a more restricted family of $Q$ distributions which has the property that the lower bound (2) can be evaluated and optimized efficiently and yet which is still sufficiently flexible as to give a good approximation to the true posterior distribution.

## 2.1 Factorized Distributions

For the purposes of building VIBES we have focussed attention initially on distributions that factorize with respect to disjoint groups $X_i$ of variables

$$Q(X|V) = \prod_i Q_i(X_i). \tag{4}$$

This approximation has been successfully used in many applications of variational methods [4, 5, 6]. Substituting (4) into (2) we can maximize $\mathcal{L}(Q)$ variationally with respect to $Q_i(X_i)$ keeping all $Q_j$ for $j \neq i$ fixed. This leads to the solution

$$\ln Q_i^\star(X_i) = \langle \ln P(V, X) \rangle_{\{j \neq i\}} + \text{const.} \tag{5}$$

where $\langle \cdot \rangle_k$ denotes an expectation with respect to the distribution $Q_k(X_k)$. Taking exponentials of both sides and normalizing we obtain

$$Q_i^\star(X_i) = \frac{\exp\langle \ln P(V, X) \rangle_{\{j \neq i\}}}{\sum_{X_i} \exp\langle \ln P(V, X) \rangle_{\{j \neq i\}}}. \tag{6}$$

Note that these are coupled equations since the solution for each $Q_i(X_i)$ depends on expectations with respect to the other factors $Q_{j \neq i}$. The variational optimization proceeds by initializing each of the $Q_i(X_i)$ and then cycling through each factor in turn replacing the current distribution with a revised estimate given by (6). The current version of VIBES is based on a factorization of the form (4) in which each factor $Q_i(X_i)$ corresponds to one of the nodes of the graph (each of which can be a composite node, as discussed shortly).

An important property of the variational update equations, from the point of view of VIBES, is that the right hand side of (6) does not depend on all of the conditional distributions $P(W_i|\text{pa}_i)$ that define the joint distribution but only on those that have a functional dependence on $X_i$, namely the conditional $P(X_i|\text{pa}_i)$, together with the conditional distributions for any children of node $i$ since these have $X_i$ in their parent set. Thus the expectations that must be performed on the right hand side of (6) involve only those variables lying in the Markov blanket of node $i$, in other words the parents, children and co-parents of $i$, as illustrated in Figure 1(a). This is a key concept in VIBES since it allows the variational update equations to be expressed in terms of local operations, which can therefore be expressed in terms of generic code which is independent of the global structure of the graph.

## 2.2 Conjugate Exponential Models

It has already been noted [4, 5] that important simplifications to the variational update equations occur when the distributions of the latent variables, conditioned on their parameters, are drawn from the exponential family and are conjugate with respect to the prior distributions of the parameters. Here we adopt a somewhat different viewpoint in that we make no distinction between latent variables and model parameters. In a Bayesian setting these both correspond to unobserved stochastic variables and can be treated on an equal footing. This allows us to consider conjugacy not just between variables and their parameters, but hierarchically between all parent-child pairs in the graph.

Thus we consider models in which each conditional distribution takes the standard exponential family form

$$\ln P(X_i|Y) = \phi_i(Y)^{\mathrm{T}} u_i(X_i) + f_i(X_i) + g_i(Y) \tag{7}$$

where the vector $\phi(Y)$ is called the *natural parameter* of the distribution. Now consider a node $Z_j$ with parent $X_i$ and co-parents $\text{cp}_j^{(i)}$, as indicated in Figure 1(a).

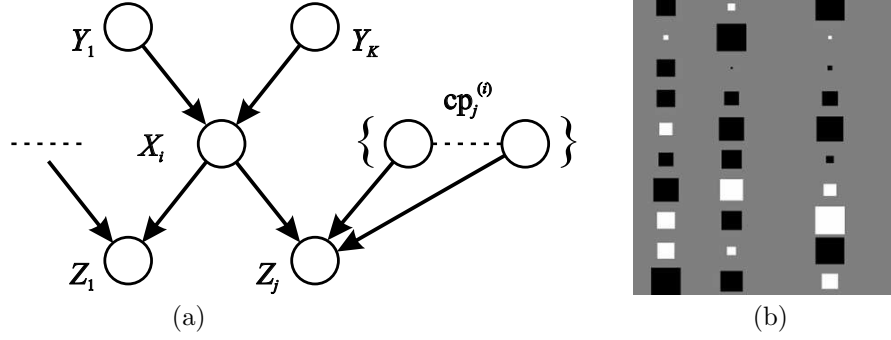

(a)                        (b)

Figure 1: (a) A central observation is that the variational update equations for node $X_i$ depend only on expectations over variables appearing in the *Markov blanket* of $X_i$, namely the set of parents, children and co-parents. (b) Hinton diagram of $\langle W \rangle$ from one of the components in the Bayesian PCA model, illustrating how all but three of the PCA eigenvectors have been suppressed.

As far as the pair of nodes $X_i$ and $Z_j$ are concerned, we can think of $P(X_i|Y)$ as a prior over $X_i$ and the conditional $P(Z_j|X_i, \mathrm{cp}_j^{(i)})$ as a (contribution to) the likelihood function. Conjugacy requires that, as a function of $X_i$, the product of these two conditionals must take the same form as (7). Since the conditional $P(Z_j|X_i, \mathrm{cp}_j^{(i)})$ is also in the exponential family it can be expressed as

$$\ln P(Z_j|X_i, \mathrm{cp}_j^{(i)}) = \phi_j(X_i, \mathrm{cp}_j^{(i)})^{\mathrm{T}} u_j(Z_j) + f_j(Z_j) + g_j(X_i, \mathrm{cp}_j^{(i)}). \qquad (8)$$

Conjugacy then requires that this be expressible in the form

$$\ln P(Z_j|X_i, \mathrm{cp}_j^{(i)}) = \widetilde{\phi}_{j \to i}(Z_j, \mathrm{cp}_j^{(i)})^{\mathrm{T}} u_i(X_i) + \lambda(Z_j, \mathrm{cp}_j^{(i)}) \qquad (9)$$

for some choice of functions $\widetilde{\phi}$ and $\lambda$. Since this must hold for each of the parents of $Z_j$ it follows that $\ln P(Z_j|X_i, \mathrm{cp}_j^{(i)})$ must be a multi-linear function of the $u_k(X_k)$ for each of the parents $X_k$ of node $XZ_j$. Also, we observe from (8) that the dependence of $\ln P(Z_j|X_i, \mathrm{cp}_j^{(i)})$ on $Z_j$ is again linear in the function $u_j(Z_j)$. We can apply a similar argument to the conjugate relationship between node $X_j$ and each of its parents, showing that the contribution from the conditional $P(X_i|Y)$ can again be expressed in terms of expectations of the natural parameters for the parent node distributions. Hence the right hand side of the variational update equation (5) for a particular node $X_i$ will be a multi-linear function of the expectations $\langle u \rangle$ for each node in the Markov blanket of $X_i$.

The variational update equation then takes the form

$$\ln Q_i^\star(X_i) = \left\{ \langle \phi_i(Y) \rangle_Y + \sum_{j=1}^{M} \langle \widetilde{\phi}_{j \to i}(Z_j, \mathrm{cp}_j^{(i)}) \rangle_{Z_j, \mathrm{cp}_j^{(i)}} \right\}^{\mathrm{T}} u_i(X_i) + \mathrm{const.} \qquad (10)$$

which involves summation of bottom up 'messages' $\langle \widetilde{\phi}_{j \to i} \rangle_{Z_j, \mathrm{cp}_j^{(i)}}$ from the children together with a top-down message $\langle \phi_i(Y) \rangle_Y$ from the parents. Since all of these messages are expressed in terms of the same basis $u_i(X_i)$, we can write compact, generic code for updating any type of node, instead of having to take account explicitly of the many possible combinations of node types in each Markov blanket.

As an example, consider the Gaussian $\mathcal{N}(X|\mu, \tau^{-1})$ for a single variable $X$ with mean $\mu$ and precision (inverse variance) $\tau$. The natural coordinates are $u_X = [X, X^2]^{\mathrm{T}}$ and the natural parameterization is $\phi = [\mu\tau, -\tau/2]^{\mathrm{T}}$. Then $\langle u \rangle = [\mu, \mu^2 + \tau^{-1}]^{\mathrm{T}}$, and the function $f_i(X_i)$ is simply zero in this case. Conjugacy allows us to choose a distribution for the parent $\mu$ that is Gaussian and a prior for $\tau$ that is a Gamma distribution. The corresponding natural parameterizations and update messages are given by

$$u_\mu = \begin{bmatrix} \mu \\ \mu^2 \end{bmatrix}, \langle \widetilde{\phi}_{X \to \mu} \rangle = \begin{bmatrix} \langle \tau \rangle \langle X \rangle \\ -\langle \tau \rangle / 2 \end{bmatrix}, u_\tau = \begin{bmatrix} \tau \\ \ln \tau \end{bmatrix}, \langle \widetilde{\phi}_{X \to \tau} \rangle = \begin{bmatrix} -\langle (X - \mu)^2 \rangle \\ 1/2 \end{bmatrix}.$$

We can similarly consider multi-dimensional Gaussian distributions, with a Gaussian prior for the mean and a Wishart prior for the inverse covariance matrix.

A generalization of the Gaussian is the rectified Gaussian which is defined as $P(X|\mu, \tau) \propto \mathcal{N}(X|\mu, \tau)$ for $X \geq 0$ and $P(X|\mu, \tau) = 0$ for $X < 0$, for which moments can be expressed in terms of the 'erf' function. This rectification corresponds to the introduction of a step function, whose logarithm corresponds to $f_i(X_i)$ in (7), which is carried through the variational update equations unchanged. Similarly, we can consider doubly truncated Gaussians, which are non-zero only over some finite interval.

Another example is the discrete distribution for categorical variables. These are most conveniently represented using the 1-of-$K$ scheme in which $S = \{S_k\}$ with $k = 1, \ldots, K$, $S_k \in \{0, 1\}$ and $\sum_k S_k = 1$. This has distribution $P(S|\pi) = \prod_{k=1}^{K} \pi_k^{S_k}$ and we can place a conjugate Dirichlet distribution over the parameters $\{\pi_k\}$.

## 2.3 Allowable Distributions

We now characterize the class of models that can be solved by VIBES using the factorized variational distribution given by (4). First of all we note that, since a Gaussian variable can have a Gaussian parent for its mean, we can extend this hierarchically to any number of levels to give a sub-graph which is a DAG of Gaussian nodes of arbitrary topology. Each Gaussian can have Gamma (or Wishart) prior over its precision.

Next, we observe that discrete variables $S = \{S_k\}$ can be used to construct 'pick' functions which choose a particular parent node $\widehat{Y}$ from amongst several conjugate parents $\{Y_k\}$, so that $\widehat{Y} = Y_k$ when $s_k = 1$, which can be written $\widehat{Y} = \prod_{k=1}^{K} Y_k^{S_k}$. Under any non-linear function $h(\cdot)$ we have $h(Y) = \prod_{k=1}^{K} h(Y_k)^{S_k}$. Furthermore the expectation under $S$ takes the form $\langle h(Y) \rangle_S = \sum_k \langle S_k \rangle h(Y_k)$. Variational inference will therefore be tractable for this model provided it is tractable for each of the parents $Y_k$ individually.

Thus we can handle the following very general architecture: an arbitrary DAG of multinomial discrete variables (each having Dirichlet priors) together with an arbitrary DAG of linear Gaussian nodes (each having Wishart priors) and with arbitrary pick links from the discrete nodes to the Gaussian nodes. This graph represents a generalization of the Gaussian mixture model, and includes as special cases models such as hidden Markov models, Kalman filters, factor analysers and principal component analysers, as well as mixtures and hierarchical mixtures of all of these.

There are other classes of models that are tractable under this scheme, for example Poisson variables having Gamma priors, although these may be of limited interest.

We can further extend the class of tractable models by considering nodes whose

natural parameters are formed from deterministic *functions* of the states of several parents. This is a key property of the VIBES approach which, as with BUGS, greatly extends its applicability. Suppose we have some conditional distribution $P(X|Y, \ldots)$ and we want to make $Y$ some deterministic function of the states of some other nodes $\psi(Z_1, \ldots, Z_M)$. In effect we have a pseudo-parent that is a deterministic function of other nodes, and indeed is represented explicitly through additional deterministic nodes in the graphical interface both to WinBUGS and to VIBES. This will be tractable under VIBES provided the expectation of $u_\psi(\psi)$ can be expressed in terms of the expectations of the corresponding functions $u_j(Z_j)$ of the parents. The pick functions discussed earlier are a special case of these deterministic functions.

Thus for a Gaussian node the mean can be formed from products and sums of the states of other Gaussian nodes provided the function is linear with respect to each of the nodes. Similarly, the precision of the Gaussian can comprise the products (but not sums) of any number of Gamma distributed variables.

Finally, we have seen that continuous nodes can have both discrete and continuous parents but that discrete nodes can only have discrete parents. We can allow discrete nodes to have continuous parents by stepping outside the conjugate-exponential framework by exploiting a variational bound on the logistic sigmoid function [1].

We also wish to be able to evaluate the lower bound (2), both to confirm the correctness of the variational updates (since the value of the bound should never decrease), as well as to monitor convergence and set termination criteria. This can be done efficiently, largely using quantities that have already been calculated during the variational updates.

## 3   VIBES: A Software Implementation

Creation of a model in VIBES simply involves drawing the graph (using operations similar to those in a simple drawing package) and then assigning properties to each node such as the functional form for the distribution, a list of the other variables it is conditioned on, and the location of the corresponding data file if the node is observed. The menu of distributions available to the user is dynamically adjusted at each stage to ensure that only valid conjugate models can be constructed.

As in WinBUGS we have adopted the convention of making logical (deterministic) nodes explicit in the graphical representation as this greatly simplifies the specification and interpretation of the model. We also use the 'plate' notation of a box surrounding one or more nodes to denote that those nodes are replicated some number of times as specified by the parameter appearing in the bottom right hand corner of the box.

### 3.1   Example: Bayesian Mixture Models

We illustrate VIBES using a Bayesian model for a mixture of $M$ probabilistic PCA distributions, each having maximum intrinsic dimensionality of $q$, with a sparse prior [6], for which the VIBES implementation is shown in Figure 2. Here there are $N$ observations of the vector $t$ whose dimensionality is $d$, as indicated by the plates. The dimensionality of the other variables is also determined by which plates they are contained in (e.g. $W$ has dimension $d \times q \times M$ whereas $\tau$ is a scalar). Variables $t$, $x$, $W$ and $\mu$ are Gaussian, $\tau$ and $\alpha$ have Gamma distributions, $S$ is discrete and $\pi$ is Dirichlet.

Once the model is completed (and the file or files containing the observed variables

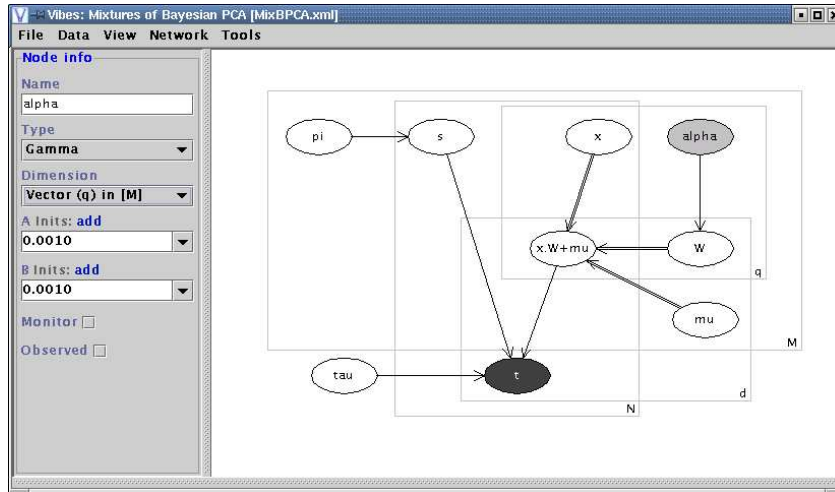

Figure 2: Screen shot from VIBES showing the graph for a mixture of probabilistic PCA distributions. The node $t$ is coloured black to denote that this variable is observed, and the node 'alpha' has been highlighted and its properties (e.g. the form of the distribution) can be changed using the menus on the left hand side. The node labelled 'x.W+mu' is a deterministic node, and the double arrows denote deterministic relationships.

are specified) it is then 'compiled', which involves allocation of memory for the variables and initializing the distributions $Q_i$ (which is done using simple heuristics but which can also be over-ridden by the user). If desired, monitoring of the lower bound (2) can be switched on (at the expense of slightly increased computation) and this can also be used to set a termination criterion. Alternatively the variational optimization can be run for a fixed number of iterations.

Once the optimization is complete various diagnostics can be used to probe the results, such as the Hinton diagram plot shown in Figure 1(b).

Now suppose we wish to modify the model, for instance by having a single set of hyper-parameters $\alpha$ whose values are shared by all of the $M$ components in the mixture, instead of having a separate set for each component. This simply involved dragging the $\alpha$ node outside of the $M$ plate using the mouse and then recompiling (since $\alpha$ is now a vector of length $q$ instead of a matrix of size $M \times q$). This literally takes a few seconds, in contrast to the effort required to formulate the variational inference equations, and develop bespoke code, for a new model! The result is then optimized as before. A screen shot of the corresponding VIBES model is shown in Figure 3.

## 4  Discussion

Our early experiences with VIBES have shown that it dramatically simplifies the construction and testing of new variational models, and readily allows a range of alternative models to be evaluated on a given problem. Currently we are extending VIBES to cater for a broader range of variational distributions by allowing the user to specify a $Q$ distribution defined over a subgraph of the true graph [7].

Finally, there are many possible extensions to the basic VIBES we have described

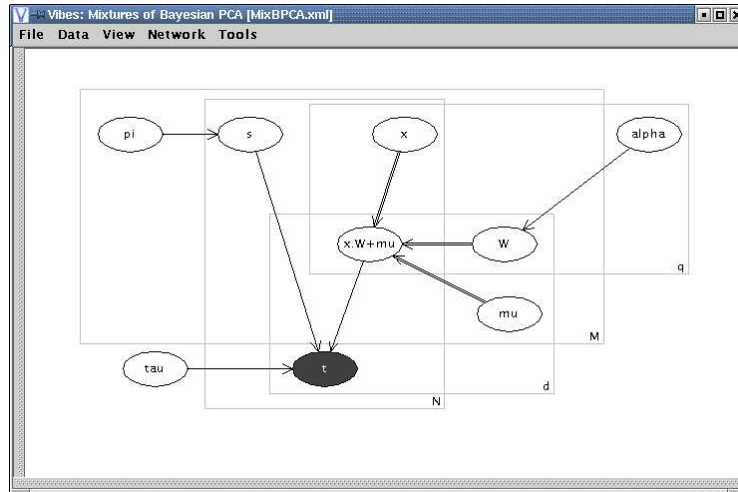

Figure 3: As in Figure 2 but with the vector $\alpha$ of hyper-parameters moved outside the $M$ 'plate'. This causes there to be only $q$ terms in $\alpha$ which are shared over the mixture components rather than $M \times q$. Note that, with no nodes highlighted, the side menus disappear.

here. For example, in order to broaden the range of models that can be tackled we can combine variational with other methods such as Gibbs sampling or optimization (empirical Bayes) to allow for non-conjugate hyper-priors for instance. Similarly, there is scope for exploiting exact methods where there exist tractable sub-graphs.

# References

[1] M. I. Jordan, Z. Ghahramani, T. S. Jaakkola, and L. K. Saul. An introduction to variational methods for graphical models. In M. I. Jordan, editor, *Learning in Graphical Models*, pages 105–162. Kluwer, 1998.

[2] R. M. Neal and G. E. Hinton. A new view of the EM algorithm that justifies incremental and other variants. In M. I. Jordan, editor, *Learning in Graphical Models*, pages 355–368. Kluwer, 1998.

[3] D J Lunn, A Thomas, N G Best, and D J Spiegelhalter. WinBUGS – a Bayesian modelling framework: concepts, structure and extensibility. *Statistics and Computing*, 10:321–333, 2000. http://www.mrc-bsu.cam.ac.uk/bugs/.

[4] Z. Ghahramani and M. J. Beal. Propagation algorithms for variational Bayesian learning. In T. K. Leen, T. Dietterich, and V. Tresp, editors, *Advances in Neural Information Processing Systems*, volume 13, Cambridge MA, 2001. MIT Press.

[5] H. Attias. A variational Bayesian framework for graphical models. In S. Solla, T. K. Leen, and K-L Muller, editors, *Advances in Neural Information Processing Systems*, volume 12, pages 209–215, Cambridge MA, 2000. MIT Press.

[6] C. M. Bishop. Variational principal components. In *Proceedings Ninth International Conference on Artificial Neural Networks, ICANN'99*, volume 1, pages 509–514. IEE, 1999.

[7] Christopher M. Bishop and John Winn. Structured variational distributions in VIBES. In *Proceedings Artificial Intelligence and Statistics*, Key West, Florida, 2003. Accepted for publication.
